# Unsupervised Structure Learning of Stochastic And-Or Grammars

**Kewei Tu**      **Maria Pavlovskaia**      **Song-Chun Zhu**
Center for Vision, Cognition, Learning and Art
Departments of Statistics and Computer Science
University of California, Los Angeles
`{tukw,mariapavl,sczhu}@ucla.edu`

## Abstract

Stochastic And-Or grammars compactly represent both compositionality and re-configurability and have been used to model different types of data such as images and events. We present a unified formalization of stochastic And-Or grammars that is agnostic to the type of the data being modeled, and propose an unsupervised approach to learning the structures as well as the parameters of such grammars. Starting from a trivial initial grammar, our approach iteratively induces compositions and reconfigurations in a unified manner and optimizes the posterior probability of the grammar. In our empirical evaluation, we applied our approach to learning event grammars and image grammars and achieved comparable or better performance than previous approaches.

## 1 Introduction

Stochastic grammars are traditionally used to represent natural language syntax and semantics, but they have also been extended to model other types of data like images [1, 2, 3] and events [4, 5, 6, 7]. It has been shown that stochastic grammars are powerful models of patterns that combine compositionality (i.e., a pattern can be decomposed into a certain configuration of sub-patterns) and reconfigurability (i.e., a pattern may have multiple alternative configurations). Stochastic grammars can be used to parse data samples into their compositional structures, which help solve tasks like classification, annotation and segmentation in a unified way. We study stochastic grammars in the form of *stochastic And-Or grammars* [1], which are an extension of stochastic grammars in natural language processing [8, 9] and are closely related to sum-product networks [10]. Stochastic And-Or grammars have been used to model spatial structures of objects and scenes [1, 3] as well as temporal structures of actions and events [7].

Manual specification of a stochastic grammar is typically very difficult and therefore machine learning approaches are often employed to automatically induce unknown stochastic grammars from data. In this paper we study unsupervised learning of stochastic And-Or grammars in which the training data are unannotated (e.g., images or action sequences).

The learning of a stochastic grammar involves two parts: learning the grammar rules (i.e., the structure of the grammar) and learning the rule probabilities or energy terms (i.e., the parameters of the grammar). One strategy in unsupervised learning of stochastic grammars is to manually specify a fixed grammar structure (in most cases, the full set of valid grammar rules) and try to optimize the parameters of the grammar. Many approaches of learning natural language grammars (e.g., [11, 12]) as well as some approaches of learning image grammars [10, 13] adopt this strategy. The main problem of this strategy is that in some scenarios the full set of valid grammar rules is too large for practical learning and inference, while manual specification of a compact grammar structure is challenging. For example, in an image grammar the number of possible grammar rules to decompose an image patch is exponential in the size of the patch; previous approaches restrict the valid

ways of decomposing an image patch (e.g., allowing only horizontal and vertical segmentations), which however reduces the expressive power of the image grammar.

In this paper, we propose an approach to learning both the structure and the parameters of a stochastic And-Or grammar. Our approach extends the previous work on structure learning of natural language grammars [14, 15, 16], while improves upon the recent work on structure learning of And-Or grammars of images [17] and events [18]. Starting from a trivial initial grammar, our approach iteratively inserts new fragments into the grammar to optimize its posterior probability. Most of the previous structure learning approaches learn new compositions and reconfigurations modeled in the grammar in a separate manner, which can be error-prone when the training data is scarce or ambiguous; in contrast, we induce *And-Or fragments* of the grammar, which unifies the search for new compositions and reconfigurations, making our approach more efficient and robust.

Our main contributions are as follows.

- We present a formalization of stochastic And-Or grammars that is agnostic to the types of atomic patterns and their compositions. Consequently, our learning approach is capable of learning from different types of data, e.g., text, images, events.

- Unlike some previous approaches that rely on heuristics for structure learning, we explicitly optimize the posterior probability of both the structure and the parameters of the grammar. The optimization procedure is made efficient by deriving and utilizing a set of sufficient statistics from the training data.

- We learn compositions and reconfigurations modeled in the grammar in a unified manner that is more efficient and robust to data scarcity and ambiguity than previous approaches.

- We empirically evaluated our approach in learning event grammars and image grammars and it achieved comparable or better performance than previous approaches.

## 2 Stochastic And-Or Grammars

Stochastic And-Or grammars are first proposed to model images [1] and later adapted to model events [7]. Here we provide a unified definition of stochastic And-Or grammars that is agnostic to the type of the data being modeled. We restrict ourselves to the context-free subclass of stochastic And-Or grammars, which can be seen as an extension of stochastic context-free grammars in formal language theory [8] as well as an extension of decomposable sum-product networks [10]. A stochastic context-free And-Or grammar is defined as a 5-tuple $\langle \Sigma, N, S, R, P \rangle$. $\Sigma$ is a set of terminal nodes representing atomic patterns that are not decomposable; $N$ is a set of nonterminal nodes representing decomposable patterns, which is divided into two disjoint sets: And-nodes $N^{\text{AND}}$ and Or-nodes $N^{\text{OR}}$; $S \in N$ is a start symbol that represents a complete entity; $R$ is a set of grammar rules, each of which represents the generation from a nonterminal node to a set of nonterminal or terminal nodes; $P$ is the set of probabilities assigned to the grammar rules. The set of grammar rules $R$ is divided into two disjoint sets: And-rules and Or-rules.

- An And-rule represents the decomposition of a pattern into a configuration of non-overlapping sub-patterns. It takes the form of $A \rightarrow a_1 a_2 \ldots a_n$, where $A \in N^{\text{AND}}$ is a nonterminal And-node and $a_1 a_2 \ldots a_n$ is a set of terminal or nonterminal nodes representing the sub-patterns. A set of relations are specified between the sub-patterns and between the nonterminal node $A$ and the sub-patterns, which configure how these sub-patterns form the composite pattern represented by $A$. The probability of an And-rule is specified by the energy terms defined on the relations. Note that one can specify different types of relations in different And-rules, which allows multiple types of compositions to be modeled in the same grammar.

- An Or-rule represents an alternative configuration of a composite pattern. It takes the form of $O \rightarrow a$, where $O \in N^{\text{OR}}$ is a nonterminal Or-node, and $a$ is either a terminal or a nonterminal node representing a possible configuration. The set of Or-rules with the same left-hand side can be written as $O \rightarrow a_1 | a_2 | \ldots | a_n$. The probability of an Or-rule specifies how likely the alternative configuration represented by the Or-rule is selected.

A stochastic And-Or grammar defines generative processes of valid entities, i.e., starting from an entity containing only the start symbol $S$ and recursively applying the grammar rules in $R$ to convert

Table 1: Examples of stochastic And-Or grammars

|  | Terminal node | Nonterminal node | Relations in And-rules |
|---|---|---|---|
| Natural language grammar | Word | Phrase | Deterministic "concatenating" relations |
| Event And-Or grammar [7] | Atomic action (e.g., standing, drinking) | Event or sub-event | Temporal relations (e.g., those proposed in [19]) |
| Image And-Or grammar [1] | Visual word (e.g., Gabor bases) | Image patch | Spatial relations (e.g., those specifying relative positions, rotations and scales) |

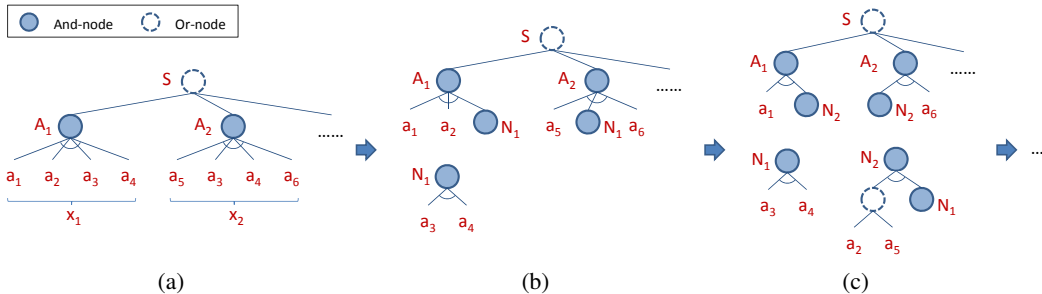

(a)             (b)             (c)

Figure 1: An illustration of the learning process. (a) The initial grammar. (b) Iteration 1: learning a grammar fragment rooted at $N_1$. (c) Iteration 2: learning a grammar fragment rooted at $N_2$.

nonterminal nodes until the entity contains only terminal nodes (atomic patterns). Table 1 gives a few examples of stochastic context-free And-Or grammars that model different types of data.

## 3 Unsupervised Structure Learning

### 3.1 Problem Definition

In unsupervised learning of stochastic And-Or grammars, we aim to learn a grammar from a set of unannotated i.i.d. data samples (e.g., natural language sentences, quantized images, action sequences). The objective function is the posterior probability of the grammar given the training data:

$$P(G|X) \propto P(G)P(X|G) = \frac{1}{Z}e^{-\alpha\|G\|}\prod_{x_i \in X} P(x_i|G)$$

where $G$ is the grammar, $X = \{x_i\}$ is the set of training samples, $Z$ is the normalization factor of the prior, $\alpha$ is a constant, and $\|G\|$ is the size of the grammar. By adopting a sparsity prior that penalizes the size of the grammar, we hope to learn a compact grammar with good generalizability. In order to ease the learning process, during learning we approximate the likelihood $P(x_i|G)$ with the Viterbi likelihood (the probability of the best parse of the data sample $x_i$). Viterbi likelihood has been empirically shown to lead to better grammar learning results [20, 10] and can be interpreted as combining the standard likelihood with an unambiguity bias [21].

### 3.2 Algorithm Framework

We first define an initial grammar that generates the exact set of training samples. Specifically, for each training sample $x_i \in X$, there is an Or-rule $S \to A_i$ in the initial grammar where $S$ is the start symbol and $A_i$ is an And-node, and the probability of the rule is $\frac{1}{\|X\|}$ where $\|X\|$ is the number of training samples; for each $x_i$ there is also an And-rule $A_i \to a_{i1}a_{i2}\ldots a_{in}$ where $a_{ij}$ $(j = 1\ldots n)$ are the terminal nodes representing the set of atomic patterns contained in sample $x_i$, and a set of relations are specified between these terminal nodes such that they compose sample $x_i$. Figure 1(a) shows an example initial grammar. This initial grammar leads to the maximal likelihood on the training data but has a very small prior probability because of its large size.

Starting from the initial grammar, we introduce new intermediate nonterminal nodes between the terminal nodes and the top-level nonterminal nodes in an iterative bottom-up fashion to generalize the grammar and increase its posterior probability. At each iteration, we add a grammar fragment into the grammar that is rooted at a new nonterminal node and contains a set of grammar rules that specify how the new nonterminal node generates one or more configurations of existing terminal or nonterminal nodes; we also try to reduce each training sample using the new grammar rules and update the top-level And-rules accordingly. Figure 1 illustrates this learning process. There are typically multiple candidate grammar fragments that can be added at each iteration, and we employ greedy search or beam search to explore the search space and maximize the posterior probability of the grammar. We also restrict the types of grammar fragments that can be added in order to reduce the number of candidate grammar fragments, which will be discussed in the next subsection. The algorithm terminates when no more grammar fragment can be found that increases the posterior probability of the grammar.

## 3.3 And-Or Fragments

In each iteration of our learning algorithm framework, we search for a new grammar fragment and add it into the grammar. There are many different types of grammar fragments, the choice of which greatly influences the efficiency and accuracy of the learning algorithm. Two simplest types of grammar fragments are *And-fragments* and *Or-fragments*. An And-fragment contains a new And-node $A$ and an And-rule $A \rightarrow a_1 a_2 \ldots a_n$ specifying the generation from the And-node $A$ to a configuration of existing nodes $a_1 a_2 \ldots a_n$. An Or-fragment contains a new Or-node $O$ and a set of Or-rules $O \rightarrow a_1 | a_2 | \ldots | a_n$ each specifying the generation from the Or-node $O$ to an existing node $a_i$. While these two types of fragments are simple and intuitive, they both have important disadvantages if they are searched for separately in the learning algorithm. For And-fragments, when the training data is scarce, many compositions modeled by the target grammar would be missing from the training data and hence cannot be learned by searching for And-fragments alone; besides, if the search for And-fragments is not properly coupled with the search for Or-fragments, the learned grammar would become large and redundant. For Or-fragments, it can be shown that in most cases adding an Or-fragment into the grammar decreases the posterior probability of the grammar even if the target grammar does contain the Or-fragment, so in order to learn Or-rules we need more expensive search techniques than greedy or beam search employed in our algorithm; in addition, the search for Or-fragments can be error-prone if different Or-rules can generate the same node in the target grammar.

Instead of And-fragments and Or-fragments, we propose to search for *And-Or fragments* in the learning algorithm. An And-Or fragment contains a new And-node $A$, a set of new Or-nodes $O_1, O_2, \ldots, O_n$, an And-rule $A \rightarrow O_1 O_2 \ldots O_n$, and a set of Or-rules $O_i \rightarrow a_{i1} | a_{i2} | \ldots | a_{im_i}$ for each Or-node $O_i$ (where $a_{i1}, a_{i2}, \ldots, a_{im_i}$ are existing nodes of the grammar). Such an And-Or fragment can generate $\prod_{i=1}^{n} m_i$ number of configurations of existing nodes. Figure 2(a) shows an example And-Or fragment. It can be shown that by adding only And-Or fragments, our algorithm is still capable of constructing any context-free And-Or grammar. Using And-Or fragments can avoid or alleviate the problems associated with And-fragments and Or-fragments: since an And-Or fragment systematically covers multiple compositions, the data scarcity problem of And-fragments is alleviated; since And-rules and Or-rules are learned in a more unified manner, the resulting grammar is often more compact; reasonable And-Or fragments usually increase the posterior probability of the grammar, therefore easing the search procedure; finally, ambiguous Or-rules can be better distinguished since they are learned jointly with their sibling Or-nodes in the And-Or fragments.

To perform greedy search or beam search, in each iteration of our learning algorithm we need to find the And-Or fragments that lead to the highest gain in the posterior probability of the grammar. Computing the posterior gain by re-parsing the training samples can be very time-consuming if the training set or the grammar is large. Fortunately, we show that by assuming grammar unambiguity the posterior gain of adding an And-Or fragment can be formulated based on a set of sufficient statistics of the training data and is efficient to compute. Since the posterior probability is proportional to the product of the likelihood and the prior probability, the posterior gain is equal to the product of the likelihood gain and the prior gain, which we formulate separately below.

**Likelihood Gain.** Remember that in our learning algorithm when an And-Or fragment is added into the grammar, we try to reduce the training samples using the new grammar rules and update the

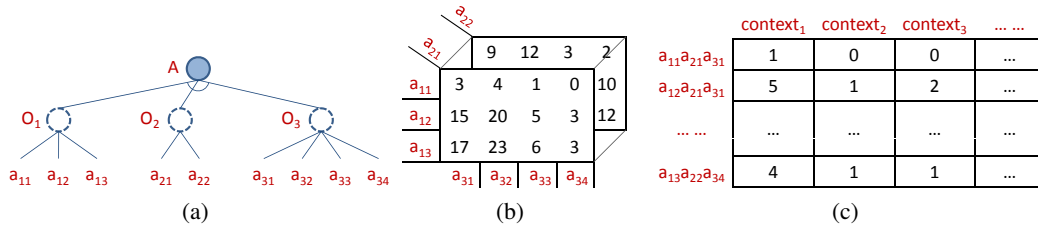

Figure 2: (a) An example And-Or fragment. (b) The $n$-gram tensor of the And-Or fragment based on the training data (here $n = 3$). (c) The context matrix of the And-Or fragment based on the training data.

top-level And-rules accordingly. Denote the set of reductions being made on the training samples by $RD$. Suppose in reduction $rd \in RD$, we replace a configuration $e$ of nodes $a_{1j_1} a_{2j_2} \ldots a_{nj_n}$ with the new And-node $A$, where $a_{ij_i} (i = 1 \ldots n)$ is an existing terminal or nonterminal node that can be generated by the new Or-node $O_i$ in the And-Or fragment. With reduction $rd$, the Viterbi likelihood of the training sample $x$ where $rd$ occurs is changed by two factors. First, since the grammar now generates the And-node $A$ first, which then generates $a_{1j_1} a_{2j_2} \ldots a_{nj_n}$, the Viterbi likelihood of sample $x$ is reduced by a factor of $P(A \rightarrow a_{1j_1} a_{2j_2} \ldots a_{nj_n})$. Second, the reduction may make sample $x$ identical to some other training samples, which increases the Viterbi likelihood of sample $x$ by a factor equal to the ratio of the numbers of such identical samples after and before the reduction. To facilitate the computation of this factor, we can construct a *context matrix* $CM$ where each row is a configuration of existing nodes covered by the And-Or fragment, each column is a context which is the surrounding patterns of a configuration, and each element is the number of times that the corresponding configuration and context co-occur in the training set. See Figure 2(c) for the context matrix of the example And-Or fragment. Putting these two types of changes to the likelihood together, we can formulate the likelihood gain of adding the And-Or fragment as follows (see the supplementary material for the full derivation).

$$\frac{P(X|G_{t+1})}{P(X|G_t)} = \frac{\prod_{i=1}^{n} \prod_{j=1}^{m_i} \|RD_i(a_{ij})\|^{\|RD_i(a_{ij})\|}}{\|RD\|^{n\|RD\|}} \times \frac{\prod_c (\sum_e CM[e,c])^{\sum_e CM[e,c]}}{\prod_{e,c} CM[e,c]^{CM[e,c]}}$$

where $G_t$ and $G_{t+1}$ are the grammars before and after learning from the And-Or fragment, $RD_i(a_{ij})$ denotes the subset of reductions in $RD$ in which the $i$-th node of the configuration being reduced is $a_{ij}$, $e$ in the summation or product ranges over all the configurations covered by the And-Or fragment, and $c$ in the product ranges over all the contexts that appear in $CM$.

It can be shown that the likelihood gain can be factorized as the product of two tensor/matrix coherence measures as defined in [22]. The first is the coherence of the $n$-gram tensor of the And-Or fragment (which tabulates the number of times each configuration covered by the And-Or fragment appears in the training samples, as illustrated in Figure 2(b)). The second is the coherence of the context matrix. These two factors provide a surrogate measure of how much the training data support the context-freeness within the And-Or fragment and the context-freeness of the And-Or fragment against its context respectively. See the supplementary material for the derivation and discussion.

The formulation of likelihood gain also entails the optimal probabilities of the Or-rules in the And-Or fragment.

$$\forall i, j \quad P(O_i \rightarrow a_{ij}) = \frac{\|RD_i(a_{ij})\|}{\sum_{j'=1}^{m_i} \|RD_i(a_{ij'})\|} = \frac{\|RD_i(a_{ij})\|}{\|RD\|}$$

**Prior Gain.** The prior probability of the grammar is determined by the grammar size. When the And-Or fragment is added into the grammar, the size of the grammar is changed in two aspects: first, the size of the grammar is increased by the size of the And-Or fragment; second, the size of the grammar is decreased because of the reductions from configurations of multiple nodes to the new And-node. Therefore, the prior gain of learning from the And-Or fragment is:

$$\frac{P(G_{t+1})}{P(G_t)} = e^{-\alpha(\|G_{t+1}\| - \|G_t\|)} = e^{-\alpha((ns_a + \sum_{i=1}^{n} m_i s_o) - \|RD\|(n-1)s_a)}$$

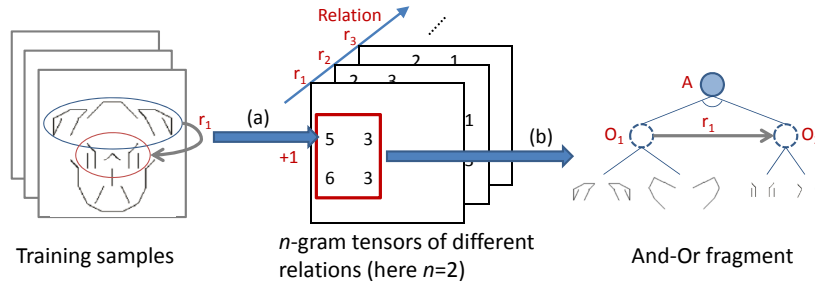

Figure 3: An illustration of the procedure of finding the best And-Or fragment. $r_1, r_2, r_3$ denote different relations between patterns. (a) Collecting statistics from the training samples to construct or update the $n$-gram tensors. (b) Finding one or more sub-tensors that lead to the highest posterior gain and constructing the corresponding And-Or fragments.

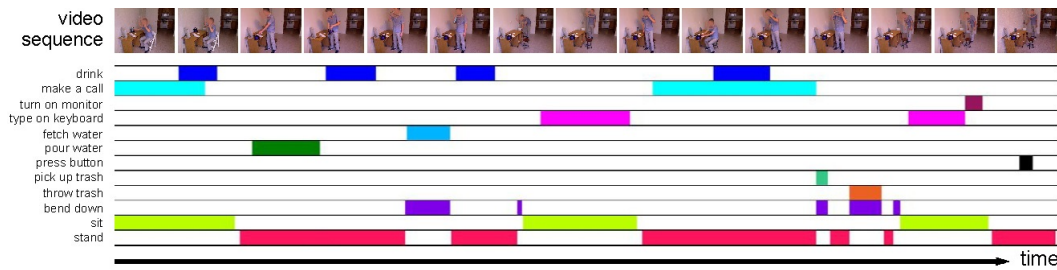

Figure 4: An example video and the action annotations from the human activity dataset [23]. Each colored bar denotes the start/end time of an occurrence of an action.

where $s_a$ and $s_o$ are the number of bits needed to encode each node on the right-hand side of an And-rule and Or-rule respectively. It can be seen that the prior gain penalizes And-Or fragments that have a large size but only cover a small number of configurations in the training data.

In order to find the And-Or fragments with the highest posterior gain, we could construct $n$-gram tensors from all the training samples for different values of $n$ and different And-rule relations, and within these $n$-gram tensors we search for sub-tensors that correspond to And-Or fragments with the highest posterior gain. Figure 3 illustrates this procedure. In practice, we find it sufficient to use greedy search or beam search with random restarts in identifying good And-Or fragments. See the supplementary material for the pseudocode of the complete algorithm of grammar learning. The algorithm runs reasonably fast: our prototype implementation can finish running within a few minutes on a desktop with 5000 training samples each containing more than 10 atomic patterns.

## 4 Experiments

### 4.1 Learning Event Grammars

We applied our approach to learn event grammars from human activity data. The first dataset contains 61 videos of indoor activities, e.g., using a computer and making a phone call [23]. The atomic actions and their start/end time are annotated in each video, as shown in Figure 4. Based on this dataset, we also synthesized a more complicated second dataset by dividing each of the two most frequent actions, sitting and standing, into three subtypes and assigning each occurrence of the two actions randomly to one of the subtypes. This simulates the scenarios in which the actions are detected in an unsupervised way and therefore actions of the same type may be regarded as different because of the difference in the posture or viewpoint.

We employed three different methods to apply our grammar learning approach on these two datasets. The first method is similar to that proposed in [18]. For each frame of a video in the dataset, we construct a binary vector that indicates which of the atomic actions are observed in this frame. In this way, each video is represented by a sequence of vectors. Consecutive vectors that are identical are

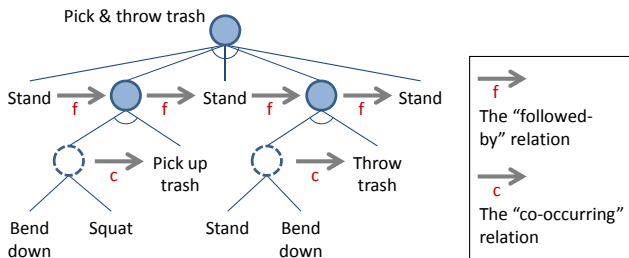

Figure 5: An example event And-Or grammar with two types of relations that grounds to atomic actions

Table 2: The experimental results (F-measure) on the event datasets. For our approach, f, c+f and cf denote the first, second and third methods respectively.

|  | Data 1 | Data 2 |
|---|---|---|
| ADIOS [15] | 0.810 | 0.204 |
| SPYZ [18] | 0.756 | 0.582 |
| Ours (f) | **0.831** | 0.702 |
| Ours (c+f) | 0.768 | 0.624 |
| Ours (cf) | 0.767 | **0.813** |

merged. We then map each distinct vector to a unique ID and thus convert each video into a sequence of IDs. Our learning approach is applied on the ID sequences, where each terminal node represents an ID and each And-node specifies the temporal "followed-by" relation between its child nodes. In the second and third methods, instead of the ID sequences, our learning approach is directly applied to the vector sequences. Each terminal node now represents an occurrence of an atomic action. In addition to the "followed-by" relation, an And-node may also specify the "co-occurring" relation between its child nodes. In this way, the resulting And-Or grammar is directly grounded to the observed atomic actions and is therefore more flexible and expressive than the grammar learned from IDs as in the first method. Figure 5 shows such a grammar. The difference between the second and the third method is: in the second method we require the And-nodes with the "co-occurring" relation to be learned before any And-node with the "followed-by" relation is learned, which is equivalent to applying the first method based on a set of IDs that are also learned; on the other hand, the third method does not restrict the order of learning of the two types of And-nodes.

Note that in our learning algorithm we assume that each training sample consists of a single pattern generated from the target grammar, but here each video may contain multiple unrelated events. We slightly modified our algorithm to accommodate this issue: right before the algorithm terminates, we change the top-level And-nodes in the grammar to Or-nodes, which removes any temporal relation between the learned events in each training sample and renders them independent of each other. When parsing a new sample using the learned grammar, we employ the CYK algorithm to efficiently identify all the subsequences that can be parsed as an event by the grammar.

We used 55 samples of each dataset as the training set and evaluated the learned grammars on the remaining 6 samples. On each testing sample, the events identified by the learned grammars were compared against manual annotations. We measured the purity (the percentage of the identified event durations overlapping with the annotated event durations) and inverse purity (the percentage of the annotated event durations overlapping with the identified event durations), and report the F-measure (the harmonic mean of purity and inverse purity). We compared our approach with two previous approaches [15, 18], both of which can only learn from ID sequences.

Table 2 shows the experimental results. It can be seen that our approach is competitive with the previous approaches on the first dataset and outperforms the previous approaches on the more complicated second dataset. Among the three methods of applying our approach, the second method has the worst performance, mostly because the restriction of learning the "co-occurring" relation first often leads to premature equating of different vectors. The third method leads to the best overall performance, which implies the advantage of grounding the grammar to atomic actions and simultaneously learning different relations. Note that the third method has better performance on the more complicated second dataset, and our analysis suggests that the division of sitting/standing into subtypes in the second dataset actually helps the third method to avoid learning erroneous compositions of continuous siting or standing.

## 4.2 Learning Image Grammars

We first tested our approach in learning image grammars from a synthetic dataset of animal face sketches [24]. Figure 6 shows some example images from the dataset. We constructed 15 training sets of 5 different sizes and ran our approach for three times on each training set. We set the terminal

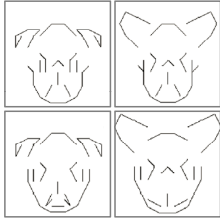

Figure 6: Example images from the synthetic dataset

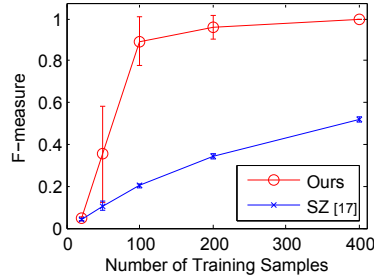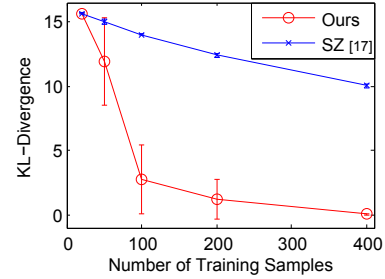

Figure 7: The experimental results on the synthetic image dataset

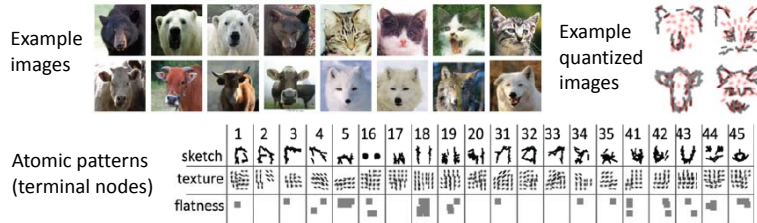

Example images

Example quantized images

Atomic patterns (terminal nodes)

Figure 8: Example images and atomic patterns of the real dataset [17]

Table 3: The average perplexity on the testing sets from the real image experiments (lower is better)

|  | Perplexity |
|---|---|
| Ours | 67.5 |
| SZ [17] | 129.4 |

nodes to represent the atomic sketches in the images and set the relations in And-rules to represent relative positions between image patches. The hyperparameter $\alpha$ of our approach is fixed to 0.5. We evaluated the learned grammars against the true grammar. We estimated the precision and recall of the sets of images generated from the learned grammars versus the true grammar, from which we computed the F-measure. We also estimated the KL-divergence of the probability distributions defined by the learned grammars from that of the true grammar. We compared our approach with the image grammar learning approach proposed in [17]. Figure 7 shows the experimental results. It can be seen that our approach significantly outperforms the competing approach.

We then ran our approach on a real dataset of animal faces that was used in [17]. The dataset contains 320 images of four categories of animals: bear, cat, cow and wolf. We followed the method described in [17] to quantize the images and learn the atomic patterns, which become the terminal nodes of the grammar. Figure 8 shows some images from the dataset, the quantization examples and the atomic patterns learned. We again used the relative positions between image patches as the type of relations in And-rules. Since the true grammar is unknown, we evaluated the learned grammars by measuring their perplexity (the reciprocal of the geometric mean probability of a sample from a testing set). We ran 10-fold cross-validation on the dataset: learning an image grammar from each training set and then evaluating its perplexity on the testing set. Before estimating the perplexity, the probability distribution represented by each learned grammar was smoothed to avoid zero probability on the testing images. Table 3 shows the results of our approach and the approach from [17]. Once again our approach significantly outperforms the competing approach.

## 5 Conclusion

We have presented a unified formalization of stochastic And-Or grammars that is agnostic to the type of the data being modeled, and have proposed an unsupervised approach to learning the structures as well as the parameters of such grammars. Our approach optimizes the posterior probability of the grammar and induces compositions and reconfigurations in a unified manner. Our experiments in learning event grammars and image grammars show satisfactory performance of our approach.

## Acknowledgments

The work is supported by grants from DARPA MSEE project FA 8650-11-1-7149, ONR MURI N00014-10-1-0933, NSF CNS 1028381, and NSF IIS 1018751.

# References

[1] S.-C. Zhu and D. Mumford, "A stochastic grammar of images," *Found. Trends. Comput. Graph. Vis.*, vol. 2, no. 4, pp. 259–362, 2006.

[2] Y. Jin and S. Geman, "Context and hierarchy in a probabilistic image model," in *CVPR*, 2006.

[3] Y. Zhao and S. C. Zhu, "Image parsing with stochastic scene grammar," in *NIPS*, 2011.

[4] Y. A. Ivanov and A. F. Bobick, "Recognition of visual activities and interactions by stochastic parsing," *Pattern Analysis and Machine Intelligence, IEEE Transactions on*, vol. 22, no. 8, pp. 852–872, 2000.

[5] M. S. Ryoo and J. K. Aggarwal, "Recognition of composite human activities through context-free grammar based representation," in *CVPR*, 2006.

[6] Z. Zhang, T. Tan, and K. Huang, "An extended grammar system for learning and recognizing complex visual events," *IEEE Trans. Pattern Anal. Mach. Intell.*, vol. 33, no. 2, pp. 240–255, Feb. 2011.

[7] M. Pei, Y. Jia, and S.-C. Zhu, "Parsing video events with goal inference and intent prediction," in *ICCV*, 2011.

[8] C. D. Manning and H. Schütze, *Foundations of statistical natural language processing*. Cambridge, MA, USA: MIT Press, 1999.

[9] P. Liang, M. I. Jordan, and D. Klein, "Probabilistic grammars and hierarchical dirichlet processes," *The handbook of applied Bayesian analysis*, 2009.

[10] H. Poon and P. Domingos, "Sum-product networks : A new deep architecture," in *Proceedings of the Twenty-Seventh Conference on Uncertainty in Artificial Intelligence (UAI)*, 2011.

[11] J. K. Baker, "Trainable grammars for speech recognition," in *Speech Communication Papers for the 97th Meeting of the Acoustical Society of America*, 1979.

[12] D. Klein and C. D. Manning, "Corpus-based induction of syntactic structure: Models of dependency and constituency," in *Proceedings of ACL*, 2004.

[13] S. Wang, Y. Wang, and S.-C. Zhu, "Hierarchical space tiling for scene modeling," in *Computer Vision–ACCV 2012*. Springer, 2013, pp. 796–810.

[14] A. Stolcke and S. M. Omohundro, "Inducing probabilistic grammars by Bayesian model merging," in *ICGI*, 1994, pp. 106–118.

[15] Z. Solan, D. Horn, E. Ruppin, and S. Edelman, "Unsupervised learning of natural languages," *Proc. Natl. Acad. Sci.*, vol. 102, no. 33, pp. 11 629–11 634, August 2005.

[16] K. Tu and V. Honavar, "Unsupervised learning of probabilistic context-free grammar using iterative bi-clustering," in *Proceedings of 9th International Colloquium on Grammatical Inference (ICGI 2008)*, ser. LNCS 5278, 2008.

[17] Z. Si and S. Zhu, "Learning and-or templates for object modeling and recognition," *IEEE Trans on Pattern Analysis and Machine Intelligence*, 2013.

[18] Z. Si, M. Pei, B. Yao, and S.-C. Zhu, "Unsupervised learning of event and-or grammar and semantics from video," in *ICCV*, 2011.

[19] J. F. Allen, "Towards a general theory of action and time," *Artificial intelligence*, vol. 23, no. 2, pp. 123–154, 1984.

[20] V. I. Spitkovsky, H. Alshawi, D. Jurafsky, and C. D. Manning, "Viterbi training improves unsupervised dependency parsing," in *Proceedings of the Fourteenth Conference on Computational Natural Language Learning*, ser. CoNLL '10, 2010.

[21] K. Tu and V. Honavar, "Unambiguity regularization for unsupervised learning of probabilistic grammars," in *Proceedings of the 2012 Conference on Empirical Methods in Natural Language Processing and Natural Language Learning (EMNLP-CoNLL 2012)*, 2012.

[22] S. C. Madeira and A. L. Oliveira, "Biclustering algorithms for biological data analysis: A survey." *IEEE/ACM Trans. on Comp. Biol. and Bioinformatics*, vol. 1, no. 1, pp. 24–45, 2004.

[23] P. Wei, N. Zheng, Y. Zhao, and S.-C. Zhu, "Concurrent action detection with structural prediction," in *Proc. Intl Conference on Computer Vision (ICCV)*, 2013.

[24] A. Barbu, M. Pavlovskaia, and S. Zhu, "Rates for inductive learning of compositional models," in *AAAI Workshop on Learning Rich Representations from Low-Level Sensors (RepLearning)*, 2013.

